# Improved Output Coding for Classification Using Continuous Relaxation

**Koby Crammer**   and   **Yoram Singer**
School of Computer Science & Engineering
The Hebrew University, Jerusalem 91904, Israel
{kobics,singer}@cs.huji.ac.il

## Abstract

Output coding is a general method for solving multiclass problems by reducing them to multiple binary classification problems. Previous research on output coding has employed, almost solely, predefined *discrete* codes. We describe an algorithm that improves the performance of output codes by relaxing them to continuous codes. The relaxation procedure is cast as an optimization problem and is reminiscent of the quadratic program for support vector machines. We describe experiments with the proposed algorithm, comparing it to standard discrete output codes. The experimental results indicate that continuous relaxations of output codes often improve the generalization performance, especially for short codes.

## 1   Introduction

The problem of multiclass categorization is about assigning labels to instances where the labels are drawn from some finite set. Many machine learning problems include a multiclass categorization component in them. Examples for such applications are text classification, optical character recognition, medical analysis, and object recognition in machine vision. There are many algorithms for the binary class problem, where there are only two possible labels, such as SVM [17], CART [4] and C4.5 [14]. Some of them can be extended to handle multiclass problems. An alternative and general approach is to reduce a multiclass problem to a multiple binary problems.

In [9] Dietterich and Bakiri described a method for reducing multiclass problems to multiple binary problems based on error correcting output codes (ECOC). Their method consists of two stages. In the training stage, a set of binary classifiers is constructed, where each classifier is trained to distinguish between two disjoint subsets of the labels. In the classification stage, each of the trained binary classifiers is applied to test instances and a voting scheme is used to decide on the label. Experimental work has shown that the output coding approach can improve performance in a wide range of problems such as text classification [3], text to speech synthesis [8], cloud classification [1] and others [9, 10, 15]. The performance of output coding was also analyzed in statistics and learning theoretic contexts [11, 12, 16, 2].

Most of previous work on output coding has concentrated on the problem of solving multiclass problems using *predefined* output codes, independently of the specific application and the learning algorithm used to construct the binary classifiers. Furthermore, the "decoding"

scheme assigns the same weight to each learned binary classifier, regardless of its performance. Last, the induced binary problems are treated as separate problems and are learned independently. Thus, there might be strong statistical correlations between the resulting classifiers, especially when the induced binary problems are similar. These problems call for an improved output coding scheme.

In a recent theoretical work [7] we suggested a relaxation of discrete output codes to *continuous* codes where each entry of the code matrix is a real number. As in discrete codes, each column of the code matrix defines a partition of the set of the labels into two subsets which are labeled positive (+) and negative (−). The sign of each entry in the code matrix determines the subset association (+ or −) and magnitude corresponds to the confidence in this association. In this paper we discuss the usage of continuous codes for multiclass problems using a two phase approach. First, we create a *binary* output code matrix that is used to train binary classifiers in the same way suggested by Dietterich and Bakiri. Given the trained classifiers and some training data we look for a more suitable continuous code by casting the problem as a constrained optimization problem. We then replace the original binary code with the improved continuous code and proceed analogously to classify new test instances.

An important property of our algorithm is that the resulting continuous code can be expressed as a linear combination of a subset of the training patterns. Since classification of new instances is performed using scalar products between the predictions vector of the binary classifiers and the rows of the code matrix, we can exploit this particular form of the code matrix and use kernels [17] to construct high dimensional product spaces. This approach enables an efficient and simple way to take into account correlations between the different binary classifiers.

The rest of this paper is organized as follows. In the next section we formally describe the framework that uses output coding for multiclass problems. In Sec. 3 we describe our algorithm for designing a continuous code from a set of binary classifiers. We describe and discuss experiments with the proposed approach in Sec. 4 and conclude in Sec. 5.

## 2 Multiclass learning using output coding

Let $S = \{(x_1, y_1), \ldots, (x_m, y_m)\}$ be a set of $m$ training examples where each instance $x_i$ belongs to a domain $\mathcal{X}$. We assume without loss of generality that each label $y_i$ is an integer from the set $\mathcal{Y} = \{1, \ldots, k\}$. A multiclass classifier is a function $H : \mathcal{X} \to \mathcal{Y}$ that maps an instance $x$ into an element $y$ of $\mathcal{Y}$. In this work we focus on a framework that uses *output codes* to build multiclass classifiers from binary classifiers. A binary output code $M$ is a matrix of size $k \times l$ over $\{-1, +1\}$ where each row of $M$ correspond to a class $y \in \mathcal{Y}$. Each column of $M$ defines a partition of $\mathcal{Y}$ into two disjoint sets. Binary learning algorithms are used to construct classifiers, one for each column $t$ of $M$. That is, the set of examples induced by column $t$ of $M$ is $(x_1, M_{t,y_1}), \ldots, (x_m, M_{t,y_m})$. This set is fed as training data to a learning algorithm that finds a binary classifier. In this work we assume that each binary classifier $h_t$ is of the form $h_t : \mathcal{X} \to \mathbb{R}$. This reduction yields $l$ different binary classifiers $h_1, \ldots, h_l$. We denote the vector of predictions of these classifiers on an instance $x$ by $\bar{h}(x) = (h_1(x), \ldots, h_l(x))$. We denote the $r$th row of $M$ by $\bar{M}_r$.

Given an example $x$ we predict the label $y$ for which the row $\bar{M}_y$ is the "most similar" to $\bar{h}(x)$. We use a general notion of similarity and define it through an inner-product function $K : \mathbb{R}^l \times \mathbb{R}^l \to \mathbb{R}$. The higher the value of $K(\bar{h}(x), \bar{M}_r)$ is the more confident we are that $r$ is the correct label of $x$ according to the set of classifiers $\bar{h}$. Note that this notion of similarity holds for both discrete and continuous matrices. An example of a simple

similarity function is $K(\bar{u}, \bar{v}) = \bar{u} \cdot \bar{v}$. It is easy to verify that when both the output code and the binary classifiers are over $\{-1, +1\}$ this choice of $K$ is equivalent to picking the row of $M$ which attains the minimal Hamming distance to $\bar{h}(x)$.

To summarize, the learning algorithm receives a training set $S$, a discrete output code (matrix) of size $k \times l$, and has access to a binary learning algorithm, denoted $L$. The learning algorithm $L$ is called $l$ times, once for each induced binary problem. The result of this process is a set of binary classifiers $\bar{h}(x) = (h_1(x), \ldots, h_l(x))$. These classifiers are fed, together with the *original* labels $y_1, \ldots, y_m$ to our second stage of the learning algorithm which learns a continuous code. This continuous code is then used to classify new instances by choosing the class which correspond to a row with the largest inner-product. The resulting classifier can be viewed as a two-layer neural network. The first (hidden) layer computes $h_1(x), \ldots, h_l(x)$ and the output unit predicts the final class by choosing the label $r$ which maximizes $K(\bar{h}(x), \bar{M}_r)$.

## 3   Finding an improved continuous code

We now describe our method for finding a continuous code that improves on a given ensemble of binary classifiers $\bar{h}$. We would like to note that we do not need to know the original code that was originally used to train the binary classifiers. For simplicity we use the standard scalar-product as our similarity function. We discuss at the end of this section more general similarity functions based on kernels which satisfy Mercer conditions.

The approach we take is to cast the code design problem as a constrained optimization problem. The multiclass empirical error is given by

$$\epsilon_S(M, \bar{h}) = \frac{1}{m} \sum_{i=1}^{m} [\![ H(x_i) \neq y_i ]\!] ,$$

where $[\![ \pi ]\!]$ is equal to 1 if the predicate $\pi$ holds and 0 otherwise. Borrowing the idea of soft margins [6] we replace the 0-1 multiclass error with a piece wise linear bound $\max_r \{\bar{h}(x_i) \cdot \bar{M}_r + b_{y_i, r}\} - \bar{h}(x_i) \cdot \bar{M}_{y_i}$, where $b_{i,j} = 1 - \delta_{i,j}$, i.e., it is equal 0 if $i = j$ and 1 otherwise. We now get an upper bound on the empirical loss

$$\epsilon_S(M, \bar{h}) \leq \frac{1}{m} \sum_{i=1}^{m} \left[ \max_r \{\bar{h}(x_i) \cdot \bar{M}_r + b_{y_i, r}\} - \bar{h}(x_i) \cdot \bar{M}_{y_i} \right] . \qquad (1)$$

Put another way, the correct label should have a confidence value that is larger by at least one than any of the confidences for the rest of the labels. Otherwise, we suffer loss which is linearly proportional to the difference between the confidence of the correct label and the maximum among the confidences of the other labels.

Define the $l_2$-norm of a code $M$ to be the $l_2$-norm of the vector represented by the concatenation of $M$'s rows, $\|M\|_2^2 = \|(\bar{M}_1, \ldots, \bar{M}_k)\|_2^2 = \sum_{i,j} M_{i,j}^2$ , where $\beta > 0$ is a regularization constant. We now cast the problem of finding a good code which minimizes the bound Eq. (1) as a quadratic optimization problem with "soft" constraints,

$$\min_{M, \xi} \quad \frac{1}{2} \beta \|M\|_2^2 + \sum_{i=1}^{m} \xi_i$$
$$\text{subject to :} \quad \forall i, r : \quad \xi_i + \bar{h}(x_i) \cdot \bar{M}_{y_i} - \bar{h}(x_i) \cdot \bar{M}_r \geq b_{i,r} . \qquad (2)$$

Solving the above optimization problem is done using its dual problem (details are omitted

due to lack of space). The solution of the dual problem result in the following form for $M$

$$\bar{M}_r = \sum_i \bar{h}(x_i)(\delta_{y_i,r} - \eta_{i,r}) \,, \qquad (3)$$

where $\eta_{i,r}$ are variables of the dual problem which satisfy $\forall i, r : \eta_{i,r} \geq 0$ and $\sum_r \eta_{i,r} = 1$. Eq. (3) implies that when the optimum of the objective function is achieved each row of the matrix $M$ is a linear combination of $\bar{h}(x_i)$. We thus say that example $i$ is a *support pattern* for class $r$ if the coefficient $(\delta_{y_i,r} - \eta_{i,r})$ of $\bar{h}(x_i)$ in Eq. (3) is non-zero. There are two cases for which example $i$ can be a support pattern for class $r$: The first is when $y_i = r$ and $\eta_{i,r} < 1$. The second case is when $y_i \neq r$ and $\eta_{i,r} > 0$. Put another way, fixing $i$, we can view $\eta_{i,r}$ as a distribution, $\bar{\eta}_i$, over the labels $r$. This distribution should give a high probability to the correct label $y_i$. Thus, an example $i$ "participates" in the solution for $M$ (Eq. (3)) if and only if $\bar{\eta}_i$ is *not* a point distribution concentrating on the correct label $y_i$. Since the continuous output code is constructed from the support patterns, we call our algorithm *SPOC* for Support Patterns Output Coding.

Denote by $\bar{\tau}_i = \bar{1}_{y_i} - \bar{\eta}_i$. Thus, from Eq. (3) we obtain the classifier,

$$H(x) = \arg\max_r \left\{ \bar{h}(x) \cdot \bar{M}_r \right\} = \arg\max_r \left\{ \sum_i \tau_{i,r} \left[ \bar{h}(x) \cdot \bar{h}(x_i) \right] \right\} \,. \qquad (4)$$

Note that solution as defined by Eq. (4) is composed of inner-products of the prediction vector on a new instance with the support patterns. Therefore, we can transform each prediction vector to some high dimensional inner-product space $\mathcal{Z}$ using a transformation $\bar{\phi} : \mathbb{R}^l \to \mathcal{Z}$. We thus replace the inner-product in the dual program with a general inner-product kernel $K$ that satisfies Mercer conditions [17]. From Eq. (4) we obtain the kernel-based classification rule $H(x)$,

$$H(x) = \arg\max_r \left\{ \sum_i \tau_{i,r} K \left( \bar{h}(x), \bar{h}(x_i) \right) \right\} \qquad (5)$$

The ability to use kernels as a means for calculating inner-products enables a simple and efficient way to take into account correlations between the binary classifiers. For instance, a second order polynomial of the form $(1 + \bar{u}\bar{v})^2$ correspond to a transformation to a feature space that includes all the products of pairs of binary classifiers. Therefore, the relaxation of discrete codes to continuous codes offers a partial remedy by assigning different importance weight to each binary classifier while taking into account the statistical correlations between the binary classifiers.

## 4 Experiments

In this section we describe experiments we performed comparing discrete and continuous output codes. We selected eight multiclass datasets, seven from the UCI repository[1] and the `mnist` dataset available from AT&T[2]. When a test set was provided we used the original split into training and test sets, otherwise we used 5-fold cross validation for evaluating the test error. Since we ran multiple experiments with 3 different codes, 7 kernels, and two base-learners, we used a subset of the training set for `mnist`, `letter`, and `shuttle`. We are in the process of performing experiments with the complete datasets and other datasets using a subset of the kernels. A summary of datasets is given in Table 1.

| Name | No. of Training Examples | No. of Test Examples | No. of Classes | No. of Attributes |
|---|---|---|---|---|
| satimage | 4435 | 2000 | 6 | 36 |
| shuttle | 5000 | 9000 | 7 | 9 |
| mnist | 5000 | 10000 | 10 | 784 |
| isolet | 6238 | 1559 | 26 | 6 |
| letter | 5000 | 4000 | 26 | 16 |
| vowel | 528 | 462 | 11 | 10 |
| glass | 214 | 5-fold cval | 7 | 10 |
| soybean | 307 | 376 | 19 | 35 |

Table 1: Description of the datasets used in experiments.

We tested three different types of codes: one-against-all (denoted "id"), BCH (a linear error correcting code), and random codes. For a classification problem with $k$ classes we set the random code to have about $10 \log_2(k)$ columns. We then set each entry in the matrix defining the code to be $-1$ or $+1$ uniformly at random. We used SVM as the base binary learning algorithm in two different modes: In the first mode we used the margin of the vector machine classifier as its real-valued prediction. That is, each binary classifier $h_t$ is of the form $h_t(x) = w{\cdot}x + b$ where $w$ and $b$ are the parameters of the separating hyperplane. In the second mode we thresholded the prediction of the classifiers, $h_t(x) = \text{sign}(w{\cdot}x + b)$. Thus, each binary classifier $h_t$ in this case is of the form $h_t : \mathcal{X} \to \{-1, +1\}$. For brevity, we refer to these classifiers as thresholded-SVMs. We would like to note in passing that this setting is by no means superficial as there are learning algorithms, such as RIPPER [5], that build classifiers of this type. We ran SPOC with 7 different kernels: homogeneous and non-homogeneous polynomials of degree 1,2, and 3, and radial-basis-functions (RBF).

A summary of the results is depicted in Figure 1. The figure contains four plots. Each plot show the relative test error difference between discrete and continuous codes. Formally, the height of each bar is proportional to $(\epsilon_d - \epsilon_c)/\epsilon_d$ where $\epsilon_d$ ($\epsilon_c$) is the test error when using a discrete (continuous) code. For each problem there are three bars, one for each type of code (one-against-all, BCH, and random). The datasets are plotted left to right in decreasing order with respect to the number of training examples per class. The left plots correspond to the results obtained using thresholded-SVM as the base binary classifier and right plots show the results using the real-valued predictions. For each mode we show the results of best performing kernel on each dataset (top plots) and the average (over the 7 different kernels) performance (bottom plots).

In general, the continuous output code relaxation indeed results in an improved performance over the original discrete output codes. The most significant improvements are achieved with thresholded-SVM as the base binary classifiers. On most problems all the kernels achieve some improvement. However, the best performing kernel seems to be problem dependent. Impressive improvements are achieved for datasets with a large number of training examples per class, shuttle being a notable example. For this dataset the test error is reduced from an average of over 3% when using discrete code to an average test error which is significantly lower than 1% for continuous codes. Furthermore, using a non-homogeneous polynomial of degree 3 reduces the test error rate down to 0.48%. In contrast, for the soybean dataset, which contains 307 training examples, and 19 classes, none of the kernels achieved any improvement, and often resulted in an increase in the test error. Examining the training error reveals that the greater the decrease in the training error due to the continuous code relaxation the worse the increase in the corresponding test error. This behavior indicates that SPOC overfitted the relatively small training set.

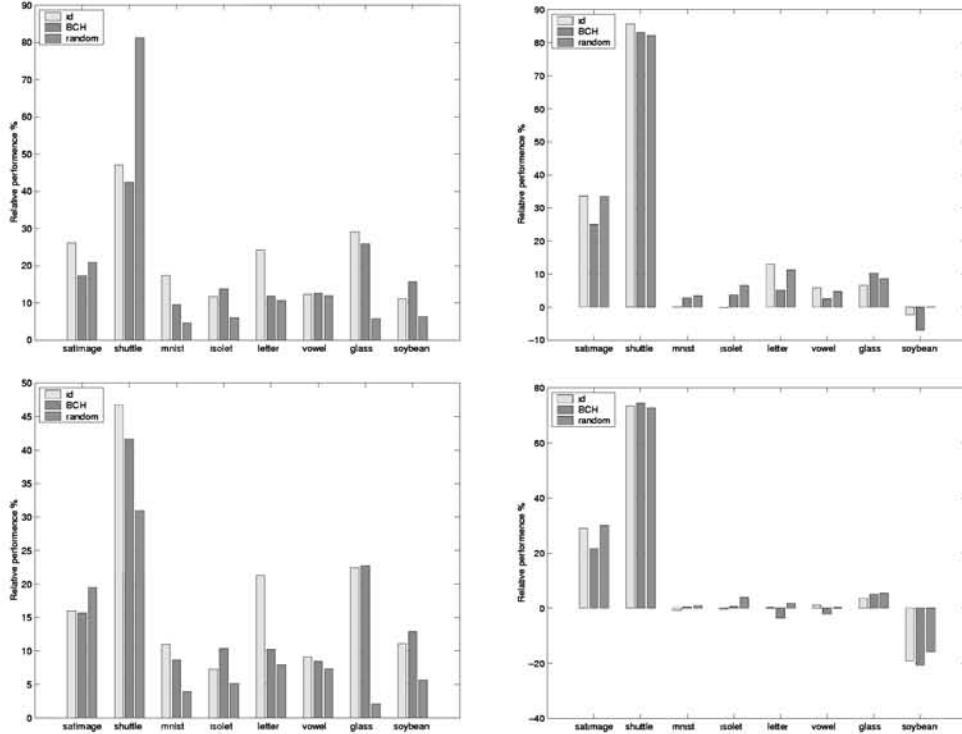

Figure 1: Comparison of the performance of discrete and continuous output codes using SVM (right figures) and thresholded-SVM (left figures) as the base learner for three different families of codes. The top figures show the relative change in test error for the best performing kernel and the bottom figures show the relative change in test error averaged across seven different kernels.

To conclude this section we describe an experiment that evaluated the performance of the SPOC algorithm as a function of the length of random codes. Using the same setting described above we ran SPOC with random codes of lengths 5 through 35 for the `vowel` dataset and lengths 15 through 50 for the `letter` dataset. In Figure 2 we show the test error rate as a function of the the code length with SVM as the base binary learner. (Similar results were obtained using thresholded-SVM as the base binary classifiers.) For the `letter` dataset we see consistent and significant improvements of the continuous codes over the discrete ones whereas for `vowel` dataset there is a major improvement for short codes that decays with the code's length. Therefore, since continuous codes can achieve performance comparable to much longer discrete codes they may serve as a viable alternative for discrete codes when computational power is limited or for classification tasks of large datasets.

## 5  Discussion

In this paper we described and experimented with an algorithm for continuous relaxation of output codes for multiclass categorization problems. The algorithm appears to be especially useful when the codes are short. An interesting question is whether the proposed approach can be generalized by calling the algorithm successively on the previous code it improved. Another viable direction is to try to combine the algorithm with other scheme for reducing

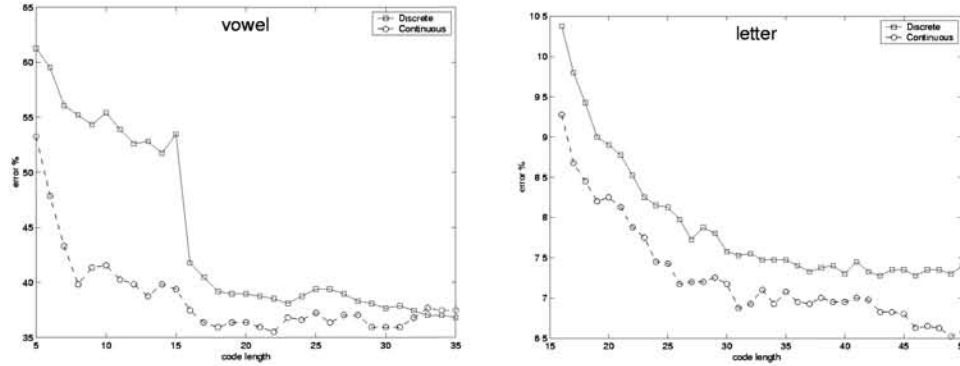

Figure 2: Comparison of the performance of discrete random codes and their continuous relaxation as a function of the code length.

multiclass problems to multiple binary problems such as tree-based codes and directed acyclic graphs [13]. We leave this for future research.

## Footnotes

[1]http://www.ics.uci.edu/~mlearn/MLRepository.html

[2]http://www.research.att.com/~yann/ocr/mnist

## References

[1] D. W. Aha and R. L. Bankert. Cloud classification using error-correcting output codes. In *Artificial Intelligence Applications: Natural Science, Agriculture, and Environmental Science*, volume 11, pages 13–28, 1997.

[2] E.L. Allwein, R.E. Schapire, and Y. Singer. Reducing multiclass to binary: A unifying approach for margin classifiers. In *Machine Learning: Proceedings of the Seventeenth International Conference*, 2000.

[3] A. Berger. Error-correcting output coding for text classification. In *IJCAI'99: Workshop on machine learning for information filtering*, 1999.

[4] Leo Breiman, Jerome H. Friedman, Richard A. Olshen, and Charles J. Stone. *Classification and Regression Trees*. Wadsworth & Brooks, 1984.

[5] William Cohen. Fast effective rule induction. In *Proceedings of the Twelfth International Conference on Machine Learning*, pages 115–123, 1995.

[6] Corinna Cortes and Vladimir Vapnik. Support-vector networks. *Machine Learning*, 20(3):273–297, September 1995.

[7] Koby Crammer and Yoram Singer. On the learnability and design of output codes for multiclass problems. In *Proceedings of the Thirteenth Annual Conference on Computational Learning Theory*, 2000.

[8] Ghulum Bakiri Thomas G. Dietterich. Achieving high-accuracy text-to-speech with machine learning. In *Data mining in speech synthesis*, 1999.

[9] Thomas G. Dietterich and Ghulum Bakiri. Solving multiclass learning problems via error-correcting output codes. *Journal of Artificial Intelligence Research*, 2:263–286, January 1995.

[10] Tom Dietterich and Eun Bae Kong. Machine learning bias, statistical bias, and statistical variance of decision tree algorithms. Technical report, Oregon State University, 1995. Available via the WWW at http://www.cs.orst.edu:80/~tgd/cv/tr.html.

[11] Trevor Hastie and Robert Tibshirani. Classification by pairwise coupling. *The Annals of Statistics*, 26(1):451–471, 1998.

[12] G. James and T. Hastie. The error coding method and PiCT. *Journal of computational and graphical stastistics*, 7(3):377–387, 1998.

[13] J.C. Platt, N. Cristianini, and J. Shawe-Taylor. Large margin dags for multiclass classification. In *Advances in Neural Information Processing Systems 12*. MIT Press, 2000. (To appear.).

[14] J. Ross Quinlan. *C4.5: Programs for Machine Learning*. Morgan Kaufmann, 1993.

[15] Robert E. Schapire. Using output codes to boost multiclass learning problems. In *Machine Learning: Proceedings of the Fourteenth International Conference*, pages 313–321, 1997.

[16] Robert E. Schapire and Yoram Singer. Improved boosting algorithms using confidence-rated predictions. *Machine Learning*, 37(3):1–40, 1999.

[17] Vladimir N. Vapnik. *Statistical Learning Theory*. Wiley, 1998.
